# Look Ma, No Hands: Analyzing the Monotonic Feature Abstraction for Text Classification

**Doug Downey**
Electrical Engineering and Computer Science Department
Northwestern University
Evanston, IL 60208
ddowney@eecs.northwestern.edu

**Oren Etzioni**
Turing Center, Department of Computer Science and Engineering
University of Washington
Seattle, WA 98195
etzioni@cs.washington.edu

## Abstract

Is accurate classification possible in the absence of hand-labeled data? This paper introduces the *Monotonic Feature* (MF) abstraction—where the probability of class membership increases monotonically with the MF's value. The paper proves that when an MF is given, PAC learning is possible with no hand-labeled data under certain assumptions.

We argue that MFs arise naturally in a broad range of textual classification applications. On the classic "20 Newsgroups" data set, a learner given an MF and unlabeled data achieves classification accuracy equal to that of a state-of-the-art semi-supervised learner relying on 160 hand-labeled examples. Even when MFs are not given as input, their presence or absence can be determined from a small amount of hand-labeled data, which yields a new semi-supervised learning method that reduces error by 15% on the 20 Newsgroups data.

## 1 Introduction

Is accurate classification possible in the complete absence of hand-labeled data? *A Priori*, the answer would seem to be no, unless the learner has knowledge of some additional problem structure. This paper identifies a problem structure, called *Monotonic Features (MFs)*, that enables the learner to automatically assign probabilistic labels to data. A feature is *monotonic* when the probability of class membership increases monotonically with that feature's value, all else being equal.

MFs occur naturally in a broad range of textual classification tasks. For example, it can be shown that Naive Bayes text classifiers return probability estimates that are monotonic in the frequency of a word—for the class in which the word is most common. Thus, if we are trying to discriminate between documents about New York and Boston, then we expect to find that the Naive Bayes feature measuring the frequency of "Giants" in the corpus is an MF for the class New York, and likewise for "Patriots" and Boston.

In document classification, the name of the class is a natural MF—the more times it is repeated in a document, all other things being equal, the more likely it is that the document belongs to the class. We demonstrate this to be the case empirically in Section 4, extending the experiments of [8] and [3]. Similarly, information retrieval systems classify documents into relevant and irrelevant documents based, in part, on the Term-Frequency-Inverse-Document-Frequency (TF-IDF) metric,

and then proceed to rank the relevant documents. The term frequency component of this metric is a monotonic feature.

The power of MFs is not restricted to textual classification. Consider Information Extraction (IE) where strings are extracted from sentences and classified into categories (*e.g.* `City` or `Film`) based on their proximity to "extraction patterns". For example, the phrase "cities such as" is an extraction pattern. Any proper noun immediately following this pattern is likely to denote a city, as in the phrase "cities such as Boston, Seattle, and New York" [9]. When classifying a proper noun, the *number of times* that it follows an extraction pattern in a corpus turns out to be a powerful MF. This observation is implicit in the combinatorial model of unsupervised IE put forth in [6]. Finally, MF-based techniques have been demonstrated to be effective for word-sense disambiguation using a set of manually-specified MFs [13]; this work was later extended to automatically derive MFs from resources like Wordnet [10].

Thus, MFs have been used *implicitly* in a broad range of textual classification tasks. This paper makes the MF abstraction *explicit*, provides a formal theory of MFs, an automatic method for explicitly detecting and utilizing MFs, and quantifies the method's benefits empirically.

## 1.1 Contribution

Typically, MFs cannot serve directly as classifiers. Instead, this paper presents theoretical and empirical results showing that even relatively weak MFs can be used to induce a noisy labeling over examples, and these examples can then be used to train effective classifiers utilizing existing supervised or semi-supervised techniques.

Our contributions are as follows:

1. We prove that the Monotonic Feature (MF) structure guarantees PAC learnability using only unlabeled data, and that MFs are distinct from and complementary to standard biases used in semi-supervised learning, including the manifold and cluster assumptions.

2. We present a general technique, called MFA, for employing MFs in combination with an arbitrary concept learning algorithm. We demonstrate experimentally that MFA can outperform state-of-the-art techniques for semi-supervised document classification, including Naive Bayes with Expectation Maximization (NB-EM), and Label Propagation, on the 20 Newsgroups data set [11].

The remainder of the paper is organized as follows. Section 2 formally defines our problem structure and the properties of monotonic features. Section 3 presents our theoretical results, and formalizes the relationship between the MF approach and previous work. We present experimental results in Section 4, and conclude with directions for future work.

## 2 Formal Framework

We consider a semi-supervised classification task, in which the goal is to produce a mapping from an instance space $\mathcal{X}$ consisting of $d$-tuples $\mathbf{x} = (x_1, \ldots, x_d)$, to a binary output space $\mathcal{Y} = \{0, 1\}$.[1] We denote the concept class of mappings $f : \mathcal{X} \to \mathcal{Y}$ as $\mathcal{C}$.

We assume the following inputs:

- A set of zero or more *labeled* examples $D_L = \{(\mathbf{x}_i, y_i) | i = 1 \ldots n\}$, drawn i.i.d. from a distribution $P(\mathbf{x}, y)$ for $\mathbf{x} \in \mathcal{X}$ and $y \in \mathcal{Y}$.
- A set of zero or more *unlabeled* examples $D_U = \{(\mathbf{x}_i) | i = 1 \ldots u\}$ drawn from the marginal distribution $P(\mathbf{x}) = \sum_y P(\mathbf{x}, y)$.
- A set $M \subset \{1, \ldots, d\}$ of zero or more *monotonic features* for the positive class $y = 1$. The monotonic features have properties specified below.

The goal of the classification task is to produce a mapping $c \in \mathcal{C}$ that maximizes classification accuracy evaluated over a set of test examples drawn i.i.d. from $P(\mathbf{x}, y)$.

We further define $\mathcal{C}_M \subset \mathcal{C}$ as the concept class of binary classifiers that use only the monotonic features. Similarly, let $\mathcal{C}_{\neg M} \subset \mathcal{C}$ indicate the concept class of binary classifiers using only the non-monotonic features.

Monotonic features exhibit a monotonically increasing relationship with the probability that an example is a member of the positive class. More formally, we define monotonic features as follows:

**Definition 1** *A **monotonic feature** for class $y$ is a feature $i \in \{1, \ldots, d\}$ for which the following three properties hold:*

- *The domain of $x_i$ is fully ordered and discrete, and has finite support.[2]*

- *The conditional probability that an example is an element of class $y = 1$ increases strictly monotonically with the value of $x_i$. That is, $P(y = 1|x_i = r) > P(y = 1|x_i = r')$ if $r > r'$.*

- *The monotonicity is non-trivial in that $P(x_i)$ has positive probability for more than one feature value. That is, there exists $r > r'$ and $\epsilon > 0$ such that $P(x_i = r), P(x_i = r') > \epsilon$.*

With this definition, we can state precisely the monotonic feature structure:

**Definition 2** *For a learning problem from the input space $\mathcal{X}$ of $d$-tuples $\mathbf{x} = (x_1, \ldots, x_d)$ to the output space $\mathcal{Y}$, the **monotonic feature structure (MFS)** holds if and only if at least one of the features $i \in \{1, \ldots, d\}$ is a monotonic feature for the positive class $y = 1$.*

When tasked with a learning problem for which the MFS holds, three distinct configurations of the input are possible. First, monotonic features may be known in the absence of labeled data ($|M| > 0, D_L = \emptyset$). This is the setting considered in previous applications of monotonic features, as discussed in the introduction. Second, monotonic features may be unknown, but labeled data may be provided ($M = \emptyset, |D_L| > 0$); this corresponds to standard semi-supervised learning. In this case, the MFS can still be exploited by identifying monotonic features using the labeled data. Lastly, both monotonic features and labeled data may be provided ($|M|, |D_L| > 0$). We provide algorithms for each case and evaluate each experimentally in Section 4.

## 3   Theory of Monotonic Features

This section shows that under certain assumptions, knowing the identity of a single monotonic feature is sufficient to PAC learn a target concept from *only unlabeled data*. Further, we prove that monotonic features become more informative relative to labeled examples as the feature set size increases. Lastly, we discuss and formally establish distinctions between the monotonic feature abstraction and other semi-supervised techniques.

We start by introducing the *conditional independence* assumption, which states that the monotonic features are conditionally independent of the non-monotonic features given the class. Formally, the conditional independence assumption is satisfied *iff* $P(\{x_i : i \in M\}|y, \{x_j : j \notin M\}) = P(\{x_i : i \in M\}|y)$. While this assumption is clearly an idealization, it is not uncommon in semi-supervised learning (for example, an analogous assumption was introduced to theoretically demonstrate the power of co-training [2]). Further , techniques based upon the idealization of conditional independence are often effective in practice (e.g., Naive Bayes Classifiers).

We show that when the concept class $\mathcal{C}_{\neg M}$ is learnable in the PAC model with classification noise, and the conditional independence assumption holds, then knowledge of a single monotonic feature makes the full concept class $\mathcal{C}$ learnable from *only* unlabeled data. Our result builds on a previous theorem from [2], and requires the following definition:

**Definition 3** *A classifier $h \in C_M$ is **weakly-useful** iff there exists $\epsilon > 0$ such that $P(h(\mathbf{x}) = 1) \geq \epsilon$ and $P(y = 1|h(\mathbf{x}) = 1) \geq P(y = 1) + \epsilon$.*

**Theorem 4** *If the conditional independence assumption is satisfied and the concept class $C_{\neg M}$ is learnable in the PAC model with classification noise, then given a single monotonic feature, $C$ is learnable from unlabeled data only.*

**Proof Sketch.** The result follows from Theorem 1 in [2] and an application of Hoeffding bounds to show that the monotonic feature can be used to construct a weakly-useful classifier.[3] □

The next theorem demonstrates that monotonic features are relatively more informative than labeled examples as the feature space increases in size. This result suggests that MF-based approaches to text classification may become increasingly valuable over time, as corpora become larger and the number of distinct words and phrases available to serve as features increases. We compare the value of monotonic features and labeled examples in terms of information gain, defined below. For convenience, these results are presented using a feature space $\mathcal{X}_B$, in which all features are binary-valued.

**Definition 5** *The **information gain** with respect to an unlabeled example's label $y$ provided by a variable $v$ is defined as the reduction in entropy of $y$ when $v$ is given, that is:*
$\sum_{y'=0,1} P(y = y'|v) \log P(y = y'|v) - P(y = y') \log P(y = y').$

Next, we define the two properties of the classification task that our theorem requires. Informally speaking, the first property states that the feature space does not have fully redundant features, whereas the second states that examples which are far apart have less dependent labels than those which are close together. We would expect these properties to hold for most tasks in practice.

**Definition 6** *A distribution $\mathcal{D}$ on $(\mathcal{X}_B, \mathcal{Y})$ has **bounded feature dependence** if there exists $\epsilon_F > 0$ such that the conditional probability $P_\mathcal{D}(x_i = r|\{x_j = r_j : j \neq i\}) < 1 - \epsilon_F$ for all $i$, $r$, and sets $\{x_j = r_j : j \neq i\}$ of assignments to one or more $x_j$.*

**Definition 7** *A distribution $\mathcal{D}$ on $(\mathcal{X}_B, \mathcal{Y})$ has **distance-diminishing information gain** if the information gain of an example $\mathbf{x}$ with respect to the label of any neighboring example $\mathbf{x}'$ is less than $K_I \delta_I^r$ for some $\delta_I < 1$, where $r$ is the Hamming distance between $\mathbf{x}$ and $\mathbf{x}'$.*

The following theorem shows that whenever the above properties hold to a sufficient degree, the expected information gain from a labeled example falls as the size of the feature space increases.

**Theorem 8** *For a learning problem governed by distribution $\mathcal{D}$ with bounded feature dependence and distance-dimishing information gain, with $\epsilon_F > \frac{\delta_I}{\delta_I + 1}$, as the number of features $d$ increases, the expected information gain provided by a labeled example about unlabeled examples' labels decreases to zero. However, the information gain from an MF $x_f$ with given relationship $P(y|x_f)$ remains constant as $d$ increases.*

The portion of Theorem 8 which concerns information gain of a labeled example is a version of the well-known "curse of dimensionality" [1], which states that the number of examples needed to estimate a function scales exponentially with the number of dimensions under certain assumptions. Theorem 8 differs in detail, however; it states the curse of dimensionality in terms of information gain, making possible a direct comparison with monotonic features.

## 3.1 Relation to Other Approaches

In the introduction, we identified several learning methods that utilized Monotonic Features (MFs) implicitly, which was a key motivation for formalizing MFs. This section explains the ways in which MF-based classification is distinct from previous semi-supervised learning methods.

When MFs are provided as input, they can be viewed as a kind of "labeled feature" studied in [7]. However, instead of a generalized expectation criteria, we use the prior to generate noisy labels for examples. Thus, MFs can complement any concept learning algorithm, not just discriminative probabilistic models as in [7]. Moreover, while [7] focuses on a problem setting in which selected

features are labeled by hand, we show in Section 4 that MFs can either obviate hand-labeled data, or can be estimated automatically from a small set of hand-labeled instances.

Co-training [2] is a semi-supervised technique that also considers a partition of the feature space into two distinct "views." One might ask if monotonic feature classification is equivalent to co-training with the monotonic features serving as one view, and the other features forming the other. However, co-training requires labeled data to train classifiers for each view, unlike monotonic feature classification which can operate without any labeled data. Thus, there are cases where an MF-based algorithm like MFA is applicable, but co-training is not.

Even when labeled data is available, co-training takes the partition of the feature set as input, whereas monotonic features can be detected automatically using the labeled data. Also, co-training is an iterative algorithm in which the most likely examples of a class according to one view are used to train a classifier on the other view in a mutually recursive fashion. For a given set of monotonic features, however, iterating this process is ineffective, because the mostly likely examples of a class according to the monotonic feature view are fixed by the monotonicity property.

### 3.1.1 Semi-supervised Smoothness Assumptions

The MFS is provably distinct from certain smoothness properties typically assumed in previous semi-supervised learning methods, known as the cluster and manifold assumptions. The *cluster assumption* states that in the target concept, the boundaries between classes occupy relatively low-density regions of the distribution $P(\mathbf{x})$. The *manifold assumption* states that the distribution $P(\mathbf{x})$ is embedded on a manifold of strictly lower dimension than the full input space $\mathcal{X}$. It can be shown that classification tasks with the MFS exist for which neither the cluster assumption nor the manifold assumption holds. Similarly, we can construct classification tasks exhibiting the manifold assumption, the cluster assumption, or their conjunction, but without the MFS. Thus, we state the following theorem.

**Theorem 9** *The monotonic feature structure neither implies nor is implied by the manifold assumption, the cluster assumption, or their conjunction or disjuntion.*

## 4 Experiments

This section reports on our experiments in utilizing MFs for text classification. As discussed in the introduction, MFs have been used *implicitly* by several classification methods in numerous tasks. Here we quantify their impact on the standard "20 newsgroups" dataset [11]. We show that MFs can be employed to perform accurate classification even without labeled examples, extending the results from [8] and [3] to a semi-supervised setting. Further, we also demonstrate that whether or not the identities of MFs are given, exploiting the MF structure by learning MFs can improve performance.

### 4.1 General Methods for Monotonic Feature Classification

Here, we define a set of abstract methods for incorporating monotonic features into any existing learning algorithm. The first method, MFA, is an abstraction of the MF word sense disambiguation algorithm first introduced in [13]. It is applicable when monotonic features are given but labeled examples are not provided. The second, MFA-SSL , applies in the standard semi-supervised learning case when some labeled examples are provided, but the identities of the MFs are unknown and must be learned. Lastly, MFA-BOTH applies when both labeled data and MF identities are given.

MFA proceeds as shown in Figure 1. MFA labels the unlabeled examples $D_U$ as elements of class $y = 1$ *iff* some monotonic feature value $x_i$ for $i \in M$ exceeds a threshold $\tau$. The threshold is set using unlabeled data so as to maximize the minimum probability mass on either side of the threshold.[4] This set of bootstrapped examples $D'_L$ is then fed as training data into a supervised or semi-supervised algorithm $\Phi(D'_L, D_U)$, and MFA outputs the resulting classifier. In general, the MFA schema can be instantiated with any concept learning algorithm $\Phi$.

```
MFA(M, D_U, Φ)
    1. D'_L = Labeled examples (x, y) such that y = 1
       iff a x_i > τ for some i ∈ M
    2. Output Φ(D'_L, D_U)
```

Figure 1: Pseudocode for MFA. The inputs are $M$, a set of monotonic features, $D_U$, a set of unlabeled examples, and $\Phi(L, U)$, a supervised or semi-supervised machine learning algorithm which outputs a classifier given labeled data $L$ and unlabeled data $U$. The threshold $\tau$ is derived from the unlabeled data and $M$ (see text).

```
MFA-SSL(D_L, D_U, Φ, Φ_M)
    1. M = the k strongest monotonic features in D_L
    2. D'_L = Examples from D_U probabilistically
       labeled with Φ_M(M, D_L, D_U)
    3. Output Φ(D_L ∪ D'_L, D_U)
```

Figure 2: Pseudocode for MFA-SSL. The inputs $D_U$ and $\Phi$ inputs are the same as those of MFA (see Figure 1). The additional inputs include labeled data $D_L$ and a machine learning algorithm $\Phi_M(M, L, U)$ which given labeled data $L$ and unlabeled data $U$ outputs a probabilistic classifier that uses only monotonic features $M$. $k$ is a parameter of MFA-SSL(see text).

When MFs are unknown, but some labeled data is given, the MFA-SSL (Figure 2) algorithm attempts to identify MFs using the labeled training data $D_L$, adding the most strongly monotonic features to the set $M$. Monotonicity strength can be measured in various ways; in our experiments, we rank each feature $x_i$ by the quantity $f(y, x_i) = \sum_r P(y, x_i = r)r$ for each class $y$.[5] MFA-SSL adds monotonic features to $M$ in descending order of this value, up to a limit of $k = 5$ per class.[6]

MFA-SSL then invokes a given machine learning algorithm $\Phi_M(M, D_L, D_U)$ to learn a probabilistic classifier that employs *only* the monotonic features in $M$. MFA-SSL uses this classifier to probabilistically label the examples in $D_U$ to form $D'_L$. MFA-SSL then returns $\Phi(D'_L, D_U)$ as in MFA. Note that when no monotonic features are identified, MFA-SSL defaults to the underlying algorithm $\Phi$.

When monotonic features are known *and* labeled examples are available, we run a derivative of MFA-SSL denoted as MFA-BOTH. The algorithm is the same as MFA-SSL, except that any given monotonic features are added to the learned set in Step 1 of Figure 2, and bootstrapped examples using the given monotonic features (from Step 1 in Figure 1) are added to $D'_L$.

## 4.2  Experimental Methodology and Baseline Methods

The task we investigate is to determine from the text of a newsgroup post the newsgroup in which it appeared. We used bag-of-word features after converting terms to lowercase, discarding the 100 most frequent terms and all terms appearing only once. Below, we present results averaged over four disjoint training sets of variable size, using a disjoint test set of 5,000 documents and an unlabeled set of 10,000 documents.

We compared the monotonic feature approach with two alternative algorithms, which represent two distinct points in the space of semi-supervised learning algorithms. The first, **NB-EM**, is a semi-supervised Naive Bayes with Expectation Maximization algorithm [12], employing settings previously shown to be effective on the 20 Newsgroups data. The second, **LP**, is a semi-supervised graph-based label propagation algorithm recently employed for text classification [4]. We found that on this dataset, the NB-EM algorithm substantially outperformed LP (providing a 41% error reduction in the experiments in Figure 3), so below we compare exclusively with NB-EM.

When the identities of monotonic features are given, we obtained one-word monotonic features simply using the newsgroup name, with minor modifications to expand abbreviations. This methodology closely followed that of [8]. For example, the occurrence count of the term "politics" was

a monotonic feature for the `talk.politics.misc` newsgroup. We also expanded the set of monotonic features to include singular/plural variants.

We employed Naive Bayes classifiers for both $\Phi$ and $\Phi_M$. We weighted the set of examples labeled using the monotonic features ($D'_L$) equally with the original labeled set, increasing the weight by the equivalent of 200 labeled examples when monotonic features are given to the algorithm.

## 4.3 Experimental Results

The first question we investigate is what level of performance MFA can achieve *without* labeled training data, when monotonic features are given. The results of this experiment are shown in Table 1. MFA achieves accuracy on the 20-way classification task of 0.563. Another way to measure this accuracy is in terms of the number of labeled examples that a baseline semi-supervised technique would require in order to achieve comparable performance. We found that MFA outperformed NB-EM with up to 160 labeled examples. This first experiment is similar to that of [8], except that instead of evaluating against only supervised techniques, we use a more comparable semi-supervised baseline (NB-EM).

Could the monotonic features, on their own, suffice to directly classify the test data? To address this question, the table also reports the performance of using the given monotonic features exclusively to label the test data (MF Alone), without using the semi-supervised technique $\Phi$. We find that the bootstrapping step provides large benefits to performance; MFA has an effective number of labeled examples eight times more than that of MF Alone.

| | Random Baseline | MF Alone | MFA |
|---|---|---|---|
| Accuracy | 5% | 24% | 56% (**2.33x**) |
| Labeled Example Equivalent | 0 | 20 | 160 (**8x**) |

Table 1: Performance of MFA when monotonic features are given, and no labeled examples are provided. MFA achieves accuracy of 0.563, which is ten fold that of a Random Baseline classifier that assigns labels randomly, and more than double that of "MF Alone", which uses only the monotonic features and ignores the other features. MFA's accuracy exceeds that of the NB-EM baseline with 160 labeled training examples, and is eight fold that of "MF Alone".

The second question we investigate is whether the monotonic feature approach can improve performance even when the class name is not given. MFA-SSL takes the same inputs as the NB-EM technique, without the identities of monotonic features. The performance of MFA-SSL as the size of the labeled data set varies is shown in Figure 3. The graph shows that for small labeled data sets of size 100-400, MFA-SSL outperforms NB-EM by an average error reduction of 15%. These results are statistically significant ($p < 0.001$, Fisher Exact Test). One important question is whether MFA-SSL's performance advantage over NB-EM is in fact due to the presence of monotonic features, or if it instead results from simply utilizing feature selection in Step 2 of Figure 2. We investigated this by replacing MFA-SSL's monotonic feature measure $f(y, x_i)$ with a standard information gain measure, and learning an equal number of features distinct from those selected by MFA-SSL originally. This method has performance essentially equivalent to that of NB-EM, suggesting that MFA-SSL's performance advantage is *not* due merely to feature selection.

Lastly, when both monotonic features and labeled examples are available, MFA-BOTH reduces error over the NB-EM baseline by an average of 31% across the training set sizes shown in Figure 3. For additional analysis of the above experiments, and results in another domain, see [5].

## 5 Conclusions

We have presented a general framework for utilizing Monotonic Features (MFs) to perform classification without hand-labeled data, or in a semi-supervised setting where monotonic features can be discovered from small numbers of hand-labeled examples. While our experiments focused on the 20 Newsgroups data set, we have complemented them with both a theoretical analysis, and by enumerating a wide variety of algorithms that have used MFs implicitly.

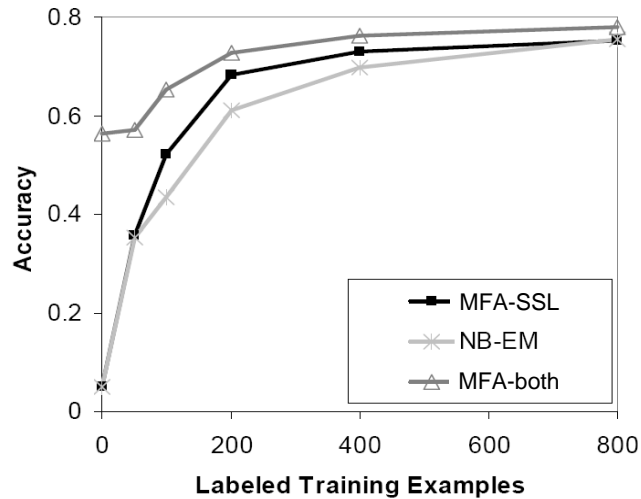

Figure 3: Performance in document classification. MFA-SSL reduces error over the NB-EM baseline by 15% for training sets between 100 and 400 examples, and MFA-BOTH reduces error by 31% overall.

## Acknowledgements

We thank Stanley Kok, Daniel Lowd, Mausam, Hoifung Poon, Alan Ritter, Stefan Schoenmackers, and Dan Weld for helpful comments. This research was supported in part by NSF grants IIS-0535284 and IIS-0312988, ONR grant N00014-08-1-0431 as well as gifts from Google, and carried out at the University of Washington's Turing Center. The first author was supported by a Microsoft Research Graduate Fellowship sponsored by Microsoft Live Labs.

## Footnotes

[1]For convenience, we restrict our formal framework to the binary case, but the techniques and analysis can be extended trivially to the multi-class case.

[2]For convenience, we present our analysis in terms of discrete and finite monotonic features, but the results can be extended naturally to the continuous case.

[3] Proofs of the theorems in this paper can be found in [5], Chapter 2.

[4]This policy is suggested by the proof of Theorem 4, in which the only requirement of the threshold is that sufficient mass lies on each side.

[5]This measure is applicable for features with numeric values. For non-numeric features, alternative measures (e.g. rank correlation) could be employed to detect MFs.

[6]A sensitivity analysis revealed that varying $k$ by up to 40% in either direction did not decrease performance of MFA-SSL in the experiments in Section 4.3.

## References

[1] R. Bellman. *Adaptive Control Processes: A Guided Tour*. Princeton University Press, 1961.

[2] A. Blum and T. Mitchell. Combining labeled and unlabeled data with co-training. In *COLT: Proceedings of the Workshop on Computational Learning Theory, Morgan Kaufmann Publishers*, pages 92–100, 1998.

[3] M.-W. Chang, L.-A. Ratinov, D. Roth, and V. Srikumar. Importance of semantic representation: Dataless classification. In D. Fox and C. P. Gomes, editors, *AAAI*, pages 830–835. AAAI Press, 2008.

[4] J. Chen, D.-H. Ji, C. L. Tan, and Z.-Y. Niu. Semi-supervised relation extraction with label propagation. In *HLT-NAACL*, 2006.

[5] D. Downey. *Redundancy in Web-scale Information Extraction: Probabilistic Model and Experimental Results*. PhD thesis, University of Washington, 2008.

[6] D. Downey, O. Etzioni, and S. Soderland. A Probabilistic Model of Redundancy in Information Extraction. In *Procs. of IJCAI*, 2005.

[7] G. Druck, G. Mann, and A. McCallum. Learning from labeled features using generalized expectation criteria. In *Proceedings of SIGIR*, 2008.

[8] A. Gliozzo, C. Strapparava, and I. Dagan. Investigating unsupervised learning for text categorization bootstrapping. In *Proceedings of HLT 2005*, pages 129–136, Morristown, NJ, USA, 2005.

[9] M. Hearst. Automatic Acquisition of Hyponyms from Large Text Corpora. In *Procs. of the 14th International Conference on Computational Linguistics*, pages 539–545, Nantes, France, 1992.

[10] R. Mihalcea and D. I. Moldovan. An automatic method for generating sense tagged corpora. In *AAAI/IAAI*, pages 461–466, 1999.

[11] T. M. Mitchell. *Machine Learning*. McGraw-Hill, New York, 1997.

[12] K. Nigam, A. McCallum, S. Thrun, and T. Mitchell. Text Classification from Labeled and Unlabeled Documents using EM. *Machine Learning*, 39(2/3):103–134, 2000.

[13] D. Yarowsky. Unsupervised word sense disambiguation rivaling supervised methods. In *Meeting of the Association for Computational Linguistics*, pages 189–196, 1995.
